# Inferring a Semantic Representation of Text via Cross-Language Correlation Analysis

**Alexei Vinokourov**
**John Shawe-Taylor**
Dept. Computer Science
Royal Holloway, University of London
Egham, Surrey, UK, TW20 0EX
alexei@cs.rhul.ac.uk
john@cs.rhul.ac.uk

**Nello Cristianini**
Dept. Statistics
UC Davis, Berkeley, US
nello@support-vector.net

## Abstract

The problem of learning a semantic representation of a text document from data is addressed, in the situation where a corpus of unlabeled paired documents is available, each pair being formed by a short English document and its French translation. This representation can then be used for any retrieval, categorization or clustering task, both in a standard and in a cross-lingual setting. By using kernel functions, in this case simple bag-of-words inner products, each part of the corpus is mapped to a high-dimensional space. The correlations between the two spaces are then learnt by using kernel Canonical Correlation Analysis. A set of directions is found in the first and in the second space that are maximally correlated. Since we assume the two representations are completely independent apart from the semantic content, any correlation between them should reflect some semantic similarity. Certain patterns of English words that relate to a specific meaning should correlate with certain patterns of French words corresponding to the same meaning, across the corpus. Using the semantic representation obtained in this way we first demonstrate that the correlations detected between the two versions of the corpus are significantly higher than random, and hence that a representation based on such features does capture statistical patterns that should reflect semantic information. Then we use such representation both in cross-language and in single-language retrieval tasks, observing performance that is consistently and significantly superior to LSI on the same data.

## 1  Introduction

Most text retrieval or categorization methods depend on exact matches between words. Such methods will, however, fail to recognize relevant documents that do not share words with a users' queries. One reason for this is that the standard representation models (e.g. boolean, standard vector, probabilistic) treat words as if they are independent, although it is clear that they are not. A central problem in this field is to automatically model term-

term semantic interrelationships, in a way to improve retrieval, and possibly to do so in an unsupervised way or with a minimal amount of supervision. For example latent semantic indexing (LSI) has been used to extract information about co-occurrence of terms in the same documents, an indicator of semantic relations, and this is achieved by singular value decomposition (SVD) of the term-document matrix. The LSI method has also been adapted to deal with the important problem of cross-language retrieval, where a query in a language is used to retrieve documents in a different language. Using a paired corpus (a set of pairs of documents, each pair being formed by two versions of the same text in two different languages), after merging each pair into a single 'document', we can interpret frequent co-occurrence of two terms in the same document as an indication of cross-linguistic correlation [5]. In this framework, a common vector-space, including words from both languages, is created and then the training set is analysed in this space using SVD. This method, termed CL-LSI, will be briefly discussed in Section 4. More generally, many other statistical and linear algebra methods have been used to obtain an improved semantic representation of text data over LSI [6]. In this study we address the problem of learning a semantic representation of text from a paired bilingual corpus, a problem that is important both for mono-lingual and cross-lingual applications. This problem can be regarded either as an unsupervised problem with paired documents, or as a supervised monolingual problem with very complex labels (i.e. the label of an english document could be its french counterpart). In either way, the data can be readily obtained without an explicit labeling effort, and furthermore there is not the loss of information due to compressing the meaning of a document into a discrete label. We employ kernel Canonical Correlation Analysis (KCCA) [1] to learn a representation of text that captures aspects of its meaning. Given a paired bilingual corpus, this method defines two embedding spaces for the documents of the corpus, one for each language, and an obvious one-to-one correspondence between points in the two spaces. KCCA then finds projections in the two embedding spaces for which the resulting projected values are highly correlated. In other words, it looks for particular combinations of words that appear to have the same co-occurrence patterns in the two languages. Our hypothesis is that finding such correlations across a paired crosslingual corpus will locate the underlying semantics, since we assume that the two languages are 'conditionally independent', or that the only thing they have in common is their meaning. The directions would carry information about the *concepts* that stood behind the process of generation of the text and, although expressed differently in different languages, are, nevertheless, semantically equivalent. To illustrate such representation we have printed the most probable (most typical) words in each language for some of the first few kernel canonical corrleation components found for bilingual $36^{th}$ Canadian Parliament corpus (Hansards) (left column is English space and right column is French space):

| PENSIONS PLAN? | | AGRICULTURE? | | CANADIAN LANDS? | | FISHING INDUSTRY? | |
|---|---|---|---|---|---|---|---|
| pension | regime | wheat | bl | park | parc | fisheries | pêches |
| plan | pensions | board | commission | land | autochtones | atlantic | atlantique |
| cpp | rpc | farmers | agriculteurs | aboriginal | terres | operatives | pêcheurs |
| canadians | prestations | newfoundland | producteurs | yukon | ches | fishermen | pêche |
| benefits | canadiens | grain | canadienne | marine | vall | newfoundland | probl |
| retirement | retraite | party | grain | government | ressources | fishery | coop |
| fund | cotisations | amendment | parti | valley | yukon | problem | ans |
| tax | fonds | producers | conseil | water | nord | operative | industrie |
| investment | discours | canadian | commercialisat | boards | gouvernement | fishing | poisson |
| income | impôt | speaker | neuve | territories | offices | industry | neuve |
| finance | revenu | referendum | ministre | board | marin | fish | terre |
| young | jeunes | minister | administration | north | eaux | years | ouest |
| years | ans | directors | modification | parks | territoires | problems | stocks |
| rate | pension | quebec | qubec | resource | parcs | wheat | ratives |
| superannuation | argent | speech | terre | agreements | nations | coast | ministre |
| disability | regimes | school | formistes | northwest | territoriales | oceans | sant |
| taxes | investissement | system | partis | resources | revendications | west | saumon |
| mounted | milliards | marketing | grains | development | ministre | salmon | affaiblies |
| future | prestation | provinces | op | treaty | cheurs | tags | facult |
| premiums | plan | constitution | nationale | nations | ouest | minister | secteur |
| seniors | finances | throne | lus | territoire | entente | communities | programme |
| country | pays | money | bloc | work | rights | program | gion |
| rates | avenir | section | nations | territory | office | commission | scientifiques |
| jobs | invalidit | rendum | chambre | atlantic | atlantique | motion | travailler |
| pay | resolution | majorit | administration | programs | ententes | stocks | conduite |

This representation is then used for retrieval tasks, providing better performance than existing techniques. Such directions are then used to calculate the coordinates of the

documents in a 'language independent' way. Of course, particular statistical care is needed for excluding 'spurious' correlations. We show that the correlations we find are not the effect of chance, and that the resulting representation significantly improves performance of retrieval systems. We find that the correlation existing between certain sets of words in English and French documents cannot be explained as a random correlation. Hence we need to explain it by means of relations between the generative processes of the two versions of the documents, that we assume to be conditionally independent given the *topic* or *content*. Under such assumptions, hence, such correlations detect similarities in content between the two documents, and can be exploited to derive a semantic representation of the text. This representation is then used for retrieval tasks, providing better performance than existing techniques. We first apply the method to crosslingual information retrieval, comparing performance with a related approach based on latent semantic indexing (LSI) described below [5]. Secondly, we treat the second language as a complex label for the first language document and view the projection obtained by CL-KCCA as a semantic map for use in a multilingual classification task with very encouraging results. From the computational point of view, we detect such correlations by solving an eigenproblem, that is avoiding problems like local minima, and we do so by using kernels.

The KCCA machinery will be given in Section 3 and in Section 4 we will show how to apply KCCA to cross-lingual retrieval while Section 4 describes the monolingual applications. Finally, results will be presented in Section 5.

## 2 Previous work

The use of LSI for cross-language retrieval was proposed by [5]. LSI uses a method from linear algebra, singular value decomposition, to discover the important associative relationships. An initial sample of documents is translated by human or, perhaps, by machine, to create a set of dual-language training documents $\{x_i\}_{i=1}^N = D_x$ and $D_y = \{y_i\}_{i=1}^N$. After preprocessing documents a common vector-space, including words from both languages, is created and then the training set is analysed in this space using SVD:

$$D = \begin{pmatrix} D_x \\ D_y \end{pmatrix} = U\Sigma V^T, \tag{1}$$

where the $i$-th column of $D$ corresponds to document $i$ with its first set of coordinates giving the first language features and the second set the second language features. To translate a new document (query) $q$ to a language-independent representation one projects (*folds-in*) its expanded (filled up with zero components related to another language) vector representation $\tilde{q}$ into the space spanned by the $k$ first eigenvectors $U_k$: $[q] = U_k^T \tilde{q}$. The similarity between two documents is measured as the inner product between their projections. The documents that are the most similar to the query are considered to be relevant.

## 3 Kernel Canonical Correlation Analysis

In this study our aim is to find an appropriate language-independent representation. Suppose as for cross-lingual LSI (CL-LSI) we are given *aligned* texts in, for simplicity, two languages, i.e., every text in one language $x_i \in \mathcal{X}$ is a translation of text $y_i \in \mathcal{Y}$ in another language or vice versa. Our hypothesis is that having the corpus $\{x_i\}_{i=1}^N$ mapped to a high-dimensional *feature* space $\mathcal{F}_x$ as $\Phi(x_i)$ and corpus $\{y_i\}_{i=1}^N$ to $\mathcal{F}_y$ as $\Phi(y_i)$ (with $K_x$ and $K_y$ being respectively the *kernels* of the two mappings, i.e. matrices of the inner products between images of all the data points, [2]), we can learn (semantic) directions $f_x \in \mathcal{F}_x$ and $f_y \in \mathcal{F}_y$ in those spaces so that the projections $(f_x, \Phi(x_i))_{i=1}^N$ and $(f_y, \Phi(y_i))_{i=1}^N$ of input data images from the different languages would be maximally correlated. We

have thus intuitively defined a need for the notion of a kernel canonical $\mathcal{F}$-correlation $\rho_{\mathcal{F}}$ ($\mathcal{F} = \mathcal{F}_x \times \mathcal{F}_y$) which is defined as

$$
\begin{aligned}
\rho_{\mathcal{F}} &= \max_{(f_x, f_y) \in \mathcal{F}} \mathrm{corr}((f_x, \Phi(x_i)), (f_y, \Phi(y_i))) \\
&= \max_{(f_x, f_y) \in \mathcal{F}} \frac{\sum_i (f_x, \Phi(x_i))(f_y, \Phi(y_i))}{\sqrt{\sum_i (f_x, \Phi(x_i))^2 \sum_j (f_y, \Phi(y_j))^2}}
\end{aligned}
\tag{2}
$$

We search for $f_x$ and $f_y$ in the space spanned by the $\Phi$-images of the data points (*reproducing kernel Hilbert space*, RKHS [2]): $f_x = \sum_l \alpha_l \Phi(x_l)$, $f_y = \sum_m \beta_m \Phi(y_m)$. This rewrites the numerator of (2) as

$$
\sum_i (f_x, \Phi(x_i))(f_y, \Phi(y_i)) = \alpha^T K_x K_y \beta
\tag{3}
$$

where $\alpha$ is the vector with components $\{\alpha_l\}$ and $\beta$ the vector with components $\{\beta_m\}$. The problem (2) can then be reformulated as

$$
\rho_{\mathcal{F}} = \max_{\alpha, \beta} \frac{\alpha^T K_x K_y \beta}{||K_x \alpha|| \, ||K_y \beta||}
\tag{4}
$$

Once we have moved to a kernel defined feature space the extra flexibility introduced means that there is a danger of overfitting. By this we mean that we can find spurious correlations by using large weight vectors to project the data so that the two projections are completely aligned. For example, if the data are linearly independent in both feature spaces we can find linear transformations that map the input data to an orthogonal basis in each feature space. It is now possible to find $N$ perfect correlations between the two representations. Using kernel functions will frequently result in linear independence of the training set, for example, when using Gaussian kernels. It is clear therefore that we will need to introduce a control on the flexibility of the projection mappings $f_x$ and $f_y$. To do that in the spirit of Partial Least Squares (PLS) we would add a multiple of 2-norm squared:

$$
\kappa ||f_x||^2 = \kappa \left( \sum_l \alpha_l \Phi(x_l), \sum_{l'} \alpha_{l'} \Phi(x_{l'}) \right) = \kappa \alpha^T K_x \alpha
\tag{5}
$$

in the denominator. Convexly combining PLS regularization term (5) and kCCA term $||K_x \alpha||^2$:

$$
(1-\kappa)||K_x \alpha||^2 + \kappa ||f_x||^2 = (1-\kappa)\alpha^T K_x^2 \alpha + \kappa \alpha^T K_x \alpha = \alpha^T K_x((1-\kappa)K_x + \kappa I)\alpha
\tag{6}
$$

we substitute its square root into denominator of (4) instead of $||K_x \alpha||$ and do the same for $\beta$:

$$
\rho_{\mathcal{F}} = \max_{\alpha, \beta} \frac{\alpha^T K_x K_y \beta}{\sqrt{((1-\kappa)||K_x \alpha||^2 + \kappa ||f_x||^2)\,((1-\kappa)||K_y \beta||^2 + \kappa ||f_y||^2)}}
\tag{7}
$$

Differentiating the expression under max with respect to $\alpha$, taking into account that $\nabla_a ||a|| = \frac{a}{||a||}$ and $\frac{\partial}{\partial \alpha} \alpha^T K_x \alpha = 2 K_x \alpha$, and equating the derivative to zero we obtain

$$
K_x K_y \beta \left( (1-\kappa)||K_x \alpha||^2 + \kappa ||f_x||^2 \right) - \alpha^T K_x K_y \beta((1-\kappa)K_x + \kappa I)K_x \alpha = 0
\tag{8}
$$

We note that $\alpha$ can be normalised so that $(1-\kappa)||K_x \alpha||^2 + \kappa ||f_x||^2 = 1$. Similar operations for $\beta$ yield analogous equations that together with (8) can be written in a matrix form:

$$
B\xi = \rho D\xi
\tag{9}
$$

where $\rho$ is the average per point correlation between projections $(f_x, \Phi(x))$ and $(f_y, \Phi(y))$: $\alpha^T K_x K_y \beta$, and

$$
B = \begin{pmatrix} O & K_x K_y \\ K_y K_x & O \end{pmatrix}, \; D = \begin{pmatrix} ((1-\kappa)K_x + \kappa I)K_x & O \\ O & ((1-\kappa)K_y + \kappa I)K_y \end{pmatrix}
\tag{10}
$$

Table 1: Statistics for 'House debates' of the $36^{th}$ Canadian Parliament proceedings corpus.

| | SENTENCE PAIRS | ENGLISH WORDS | FRENCH WORDS |
|---|---|---|---|
| TRAINING | 948K | 14,614K | 15,657K |
| TESTING 1 | 62K | 995K | 1067K |

where $\xi = \left(\alpha^T \; \beta^T\right)^T$. Equation (9) is known as a generalised eigenvalue problem.The standard approach to the solution of (9) in the case of a symmetric $D$ is to perform incomplete Cholesky decomposition of the matrix $D$: $D = C^T C$ and define $\varsigma = C\xi$ which allows us, after simple transformations, to rewrite it as a standard eigenvalue problem $C^{-T}BC^{-1}\varsigma = \rho\varsigma$. We will discuss how to choose $\kappa$ in Section 5.

It is easy to see that if $\alpha$ or $\beta$ changes sign in (9), $\rho$ also changes sign. Thus, the spectrum of the problem (9) has paired positive and negative values between $-1$ and $1$.

## 4  Applications of KCCA

**Cross-linguistic retrieval with KCCA**. The kernel CCA procedure identifies a set of projections from both languages into a common semantic space. This provides a natural framework for performing cross-language information retrieval. We first select a number $d$ of semantic dimensions, $1 \leq d \leq N$, with largest correlation values $\rho$. To process an incoming query $q$ we expand $q$ into the vector representation for its language $\widetilde{q}$ and project it onto the $d$ canonical $\mathcal{F}$-correlation components: $[q] = A^T Z^T \widetilde{q}$ using the appropriate vector for that language, where $A$ is a $N \times d$ matrix whose columns are the first solutions of (9) for the given language sorted by eigenvalue in descending order. Here we assumed that $(\Phi(z), \Phi(\widetilde{q}))$ is simply $z^T\widetilde{q}$ where $Z$ is the training corpus in the given language: $Z = (\; x_1 \quad x_2 \quad \ldots \quad x_N \;)$ or $Z = (\; y_1 \quad y_2 \quad \ldots \quad y_N \;)$.

**Using the semantic space in text categorisation**. The semantic vectors in the given language $W = ZA$ can be exported and used in some other application, for example, Support Vector Machine classification. We first find common features of the training data used to extract the semantics and the data used to train SVM classifier, cut the features that are not common and compute the new kernel which is the inner product of the projected data:

$$K'(x_i, x_j) = x_i^T WW^T x_j \tag{11}$$

The term-term relationship matrix $WW^T$ can be computed only once and stored for further use in the SVM learning process and classification.

## 5  Experiments

**Experimental setup**. Following [5] we conducted a series of experiments with the Hansard collection [3] to measure the ability of CL-LSI and CL-KCCA for any document from a test collection in one language to find its mate in another language. The whole collection consists of 1.3 million pairs of aligned text chunks (sentences or smaller fragments) from the $36^{th}$ Canadian Parliament proceedings. In our experiments we used only the 'house debates' part for which statistics is given in Table 1. As a testing collection we used only 'testing 1'. The raw text was split into sentences with Adwait Ratnaparkhi's MXTERMINATOR and the sentences were aligned with I. Dan Melamed's GSA tool (for details on the collection and also for the source see [3]).

Table 2: Average accuracy of top-rank (first retrieved) English→French retrieval, % (left) and average precision of English→French retrieval over set of fixed recalls $(0.1, 0.2, \ldots, 0.9)$, % (right)

| $d$ | 100 | 200 | 300 | 400 | full | 100 | 200 | 300 | 400 | full |
|---|---|---|---|---|---|---|---|---|---|---|
| cl-lsi | 84±3 | 91±1 | 93±1 | 95±1 | 97±1 | 73±1 | 78±1 | 80±1 | 82±1 | 82±6 |
| cl-kcca | 98±1 | 99±1 | 99±1 | 99±1 | 99±1 | 91±2 | 91±2 | 91±2 | 91±2 | 87±4 |

The text chunks were split into 'paragraphs' based on '***' delimiters and these 'paragraphs' were treated as separate documents. After removing stop-words in both French and English parts and rare words (i.e. appearing less than three times) we obtained $5159 \times 12738$ term-by-document 'English' matrix and $5611 \times 12738$ 'French' matrix (we also removed a few documents that appeared to be problematic when split into paragraphs). As these matrices were still too large to perform SVD and KCCA on them, we split the whole collection into 14 chunks of about 910 documents each and conducted experiments separately with them, measuring the performance of the methods each time on a 917-document test collection. The results were then averaged. We have also trained the CL-KCCA method on randomly reassociated French-English document pairs and observed accuracy of about 0.15 on test data which is far lower than results on the non-random original data. It is worth noting that CL-KCCA behaves differently from CL-LSI over the full scale of the spectrum. When CL-LSI only increases its performance with more eigenvectors taken from the lower part of spectrum (which is, somewhat unexpectedly, quite different from its behaviour in the monolinguistic setting), CL-KCCA's performance, on the contrary, tends to deteriorate with the dimensionality of the semantic subspace approaching the dimensionality of the input data space.

The partial Singular Value Decomposition of the matrices was done using Matlab's 'svds' function and full SVD was performed using the 'kernel trick' discussed in the previous section and 'svd' function which took about 2 minutes to compute on Linux Pentium III 1GHz system for a selection of 1000 documents. The Matlab implementation of KCCA using the same function, 'svd', which solves the generalised eigenvalue problem through Cholesky incomplete decomposition, took about 8 minutes to compute on the same data.

**Mate retrieval**. The results are presented in Table 2. Only one - mate document in French was considered as relevant to each of the test English documents which were treated as queries and the relative number of correctly retrieved documents was computed (Table 2) along with average precision over fixed recalls: 0.1, 0.2, …, 0.9. Very similar results (omitted here) were obtained when French documents were treated as queries and English as test documents. As one can see from Table 2 CL-KCCA seems to capture most of the semantics in the first few components achieving 98% accuracy with as little as 100 components when CL-LSI needs all components for a similar figure.

**Selecting the regularization parameter**. The regularization parameter $\kappa$ (6) not only makes the problem (9) well-posed numerically, but also provides control over capacity of the function space where the solution is being sought. The larger values of $\kappa$ are, the less sensitive the method to the input data is, therefore, the more stable (less prone to finding spurious relations) the solution becomes. We should thus observe an increase of "reliability" of the solution. We measure the ability of the method to catch useful signal by comparing the solutions on original input and "random" data. The "random" data is constructed by random reassociations of the data pairs, for example, $(E, \mathrm{rand}(F))$ denotes English-French parallel corpus which is obtained from the original English-French aligned collection by reshuffling the French (equivalently, English) documents. Suppose, $\boldsymbol{\rho}_\kappa = KCCA_\kappa(D_1, D_2)$ denotes the (positive part of) spectrum of the KCCA solution on the paired dataset $(D_1, D_2)$. If the method is overfitting the data it will be able to find perfect correlations and hence $\|\mathbf{j} - KCCA_\kappa(D_{21}, D_{22})\| \approx 0$, where $\mathbf{j}$ is the all-one vec-

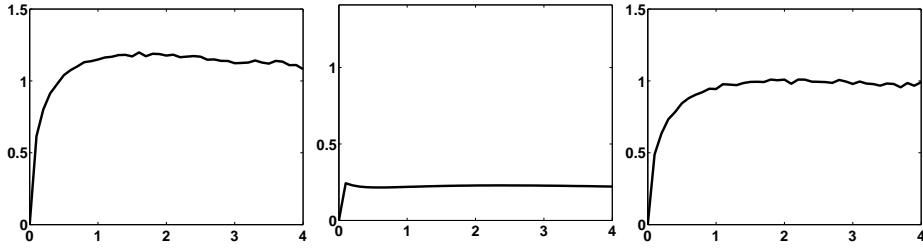

Figure 1: Quantities $||\mathbf{j} - KCCA_\kappa(E, \mathrm{rand}(E))||$ (left), $||\mathbf{j} - KCCA_\kappa(E, F)||$ (middle) and $||\mathbf{j} - KCCA_\kappa(E, \mathrm{rand}(F))||$ (right) as functions of the regularization parameter $\kappa$. (Graphs were obtained for the regularization schema discussed in [1]).

tor. We therefore use this as a measure to assess the degree of overfitting. Three graphs in Figure 1 show the quantities $||\mathbf{j}-KCCA_\kappa(E, \mathrm{rand}(E))||$, $||\mathbf{j}-KCCA_\kappa(E, F)||$, and $||\mathbf{j}-KCCA_\kappa(E, \mathrm{rand}(F))||$ as functions of the regularization parameter $\kappa$. For small values of $\kappa$ the spectrum of all the tests is close to the all-one spectrum (the spectrum $KCCA_\kappa(E, E)$). This indicates overfitting since the method is able to find correlations even in randomly associated pairs. As $\kappa$ increases the spectrum of the randomly associated data becomes far from all-one, while that of the paired documents remains correlated. This observation can be exploited for choosing the optimal value of $\kappa$. From the middle and right graphs in Figure 1 this value could be derived as lying somewhere between 1 and 2. For the experiments reported in this study we used the value of 1.5.

**Pseudo query test**. To perform a more realistic test we generated short queries, which are most likely to occur in search engines, that consisted of the 5 most probable words from each test document. The relevant documents were the test documents themselves in monolinguistic retrieval (English query - English document) and their mates in the cross-linguistic (English query - French document) test. Table 3 shows the relative number of correctly retrieved as top-ranked English documents for English queries (left) and the relative number of correctly retrieved documents in the top ten ranked (right). Table 4 provides analogous results but for cross-linguistic retrieval.

Table 3: English→ English top-ranked retrieval accuracy, % (left) and English→ English top-ten retrieval accuracy, % (right)

| $d$ | 100 | 200 | 300 | 400 | full | 100 | 200 | 300 | 400 | full |
|---|---|---|---|---|---|---|---|---|---|---|
| cl-lsi | 53±2 | 60±1 | 64±1 | 66±1 | 70±1 | 82±1 | 86±1 | 88±1 | 89±1 | 91±1 |
| cl-kcca | 60±1 | 63±1 | 70±1 | 71±1 | 73±1 | 90±1 | 93±1 | 94±1 | 95±1 | 95±1 |

Table 4: English→ French top-ranked retrieval accuracy, % (left) and English-French top-ten retrieval accuracy, % (right)

| $d$ | 100 | 200 | 300 | 400 | full | 100 | 200 | 300 | 400 | full |
|---|---|---|---|---|---|---|---|---|---|---|
| cl-lsi | 30±1 | 38±1 | 42±2 | 45±1 | 49±6 | 67±1 | 75±2 | 79±2 | 81±2 | 84±1 |
| cl-kcca | 68±1 | 75±1 | 78±1 | 79±1 | 81±1 | 94±1 | 96±1 | 97±1 | 98±1 | 98±1 |

**Text categorisation using semantics learned on a completely different corpus**. The semantics (300 vectors) extracted from the Canadian Parliament corpus (Hansard) was used in Support Vector Machine (SVM) text classification [2] of Reuters-21578 corpus (Table 5). In this experimental setting the intersection of vector spaces of the Hansards,

5159 English words from the first 1000-French-English-document training chunk, and `Reuters ModApt` split, 9962 words from the 9602 training and 3299 test documents had 1473 words. The extracted $d = 300$ KCCA vectors from English and French parts (raw 'KCCA' of Table 5) and 300 eigenvectors from the same data (raw 'CL-LSI') were used in the $\text{SVM}^{light}$ [4] with the kernel (11) to classify the `Reuters-21578` data. The experiments were averaged over 10 runs with 5% each time randomly chosen fraction of training data as the difference between bag-of-words and semantic methods is more contrasting on smaller samples. Both CL-KCCA and CL-LSI perform remarkably well when one considers that they are based on just 1473 words. In all cases CL-KCCA outperforms the bag-of-words kernel.

Table 5: $F_1$ value, %, averaged over 10 subsequent runs of SVM classifier with original `Reuters-21578` data ('bag-of-words') and preprocessed using semantics (300 vectors) extracted from the `Canadian Parliament` corpus by various methods.

| CLASS | EARN | ACQ | GRAIN | CRUDE |
|---|---|---|---|---|
| BAG-OF-WORDS | 81±7 | 57±3 | 33±5 | 13±3 |
| CL-KCCA | **90**±2 | **75**±4 | 43±6 | 38±12 |
| CL-LSI | 77±3 | 52±3 | **64**±14 | **40**±2 |

## 6 Conclusions

We have presented a novel procedure for extracting semantic information in an unsupervised way from a bilingual corpus, and we have used it in text retrieval applications. Our main findings are that: the correlation existing between certain sets of words in english and french documents cannot be explained as random correlations. Hence we need to explain it by means of relations between the generative processes of the two versions of the documents. The correlations detect similarities in content between the two documents, and can be exploited to derive a semantic representation of the text. The representation is then used for retrieval tasks, providing better performance than existing techniques.

## References

[1] F. R. Bach and M. I. Jordan. Kernel indepedendent component analysis. *Journal of Machine Learning Research*, 3:1–48, 2002.

[2] Nello Cristianini and John Shawe-Taylor. *An introduction to Support Vector Machines and other kernel-based learning methods*. Cambridge University Press, 2000.

[3] Ulrich Germann. Aligned Hansards of the 36th Parliament of Canada. http://www.isi.edu/natural-language/download/hansard/, 2001. Release 2001-1a.

[4] Thorsten Joachims. $SVM^{light}$ - Support Vector Machine. http://svmlight.joachims.org, 2002.

[5] M. L. Littman, S. T. Dumais, and T. K. Landauer. Automatic cross-language information retrieval using latent semantic indexing. In G. Grefenstette, editor, *Cross language information retrieval*. Kluwer, 1998.

[6] Alexei Vinokourov and Mark Girolami. A probabilistic framework for the hierarchic organisation and classification of document collections. *Journal of Intelligent Information Systems*, 18(2/3):153–172, 2002. Special Issue on Automated Text Categorization.
